# Topic-Partitioned Multinetwork Embeddings

**Peter Krafft**[*]
CSAIL
MIT
pkrafft@mit.edu

**Juston Moore**[†]**, Bruce Desmarais**[‡]**, Hanna Wallach**[†]
[†]Department of Computer Science, [‡]Department of Political Science
University of Massachusetts Amherst
[†]{jmoore, wallach}@cs.umass.edu
[‡]desmarais@polsci.umass.edu

## Abstract

We introduce a new Bayesian admixture model intended for exploratory analysis of communication networks—specifically, the discovery and visualization of topic-specific subnetworks in email data sets. Our model produces principled visualizations of email networks, i.e., visualizations that have precise mathematical interpretations in terms of our model and its relationship to the observed data. We validate our modeling assumptions by demonstrating that our model achieves better link prediction performance than three state-of-the-art network models and exhibits topic coherence comparable to that of latent Dirichlet allocation. We showcase our model's ability to discover and visualize topic-specific communication patterns using a new email data set: the New Hanover County email network. We provide an extensive analysis of these communication patterns, leading us to recommend our model for any exploratory analysis of email networks or other similarly-structured communication data. Finally, we advocate for principled visualization as a primary objective in the development of new network models.

## 1   Introduction

The structures of organizational communication networks are critical to collaborative problem solving [1]. Although it is seldom possible for researchers to directly observe complete organizational communication networks, email data sets provide one means by which they can at least partially observe and reason about them. As a result—and especially in light of their rich textual detail, existing infrastructure, and widespread usage—email data sets hold the potential to answer many important scientific and practical questions within the organizational and social sciences. While some questions may be answered by studying the structure of an email network as a whole, other, more nuanced, questions can only be answered at finer levels of granularity—specifically, by studying topic-specific subnetworks. For example, breaks in communication (or duplicated communication) about particular topics may indicate a need for some form of organizational restructuring. In order to facilitate the study of these kinds of questions, we present a new Bayesian admixture model intended for discovering and summarizing topic-specific communication subnetworks in email data sets.

There are a number of probabilistic models that incorporate both network and text data. Although some of these models are specifically for email networks (e.g., McCallum et al.'s author–recipient–topic model [2]), most are intended for networks of documents, such as web pages and the links between them [3] or academic papers and their citations [4]. In contrast, an email network is more naturally viewed as a network of actors exchanging documents, i.e., actors are associated with nodes while documents are associated with edges. In other words, an email network defines a multinetwork in which there may be multiple edges (one per email) between any pair of actors. Perhaps more importantly, much of the recent work on modeling networks and text has focused on tasks such as

---

[*]Work done at the University of Massachusetts Amherst

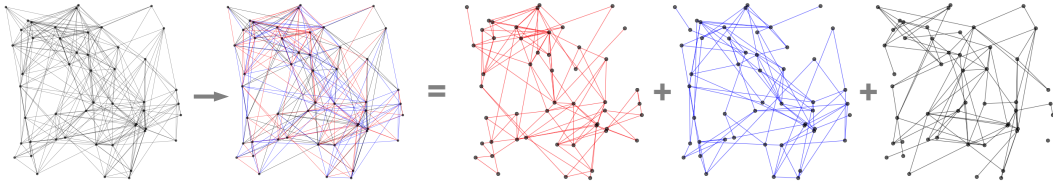

Figure 1: Our model partitions an observed email network (left) into topic-specific subnetworks (right) by associating each author–recipient edge in the observed network with a single topic.

predicting links or detecting communities. Instead, we take a complementary approach and focus on exploratory analysis. Specifically, our goal is to discover and visualize topic-specific subnetworks.

Rather than taking a two-stage approach in which subnetworks are discovered using one model and visualized using another, we present a single probabilistic model that partitions an observed email network into topic-specific subnetworks while simultaneously producing a visual representation of each subnetwork. If network modeling and visualization are undertaken separately, the resultant visualizations may not directly reflect the model and its relationship to the observed data. Rather, these visualizations provide a view of the model and the data seen through the lens of the visualization algorithm and its associated assumptions, so any conclusions drawn from such visualizations can be biased by artifacts of the visualization algorithm. Producing principled visualizations of networks, i.e., visualizations that have precise interpretations in terms of an associated network model and its relationship to the observed data, remains an open challenge in statistical network modeling [5]. Addressing this open challenge was a primary objective in the development of our new model.

In order to discover and visualize topic-specific subnetworks, our model must associate each author–recipient edge in the observed email network with a topic, as shown in Figure 1. Our model draws upon ideas from latent Dirichlet allocation (LDA) [6] to identify a set of corpus-wide topics of communication, as well as the subset of topics that best describe each observed email. We model network structure using an approach similar to that of Hoff et al.'s latent space model (LSM) [7] so as to facilitate visualization. Given an observed network, LSM associates each actor in the network with a point in $K$-dimensional Euclidean space. For any pair of actors, the smaller the distance between their points, the more likely they are to interact. If $K = 2$ or $K = 3$, these interaction probabilities, collectively known as a "communication pattern", can be directly visualized in 2- or 3-dimensional space via the locations of the actor-specific points. Our model extends this idea by associating a $K$-dimensional Euclidean space with each topic. Observed author–recipient edges are explicitly associated with topics via the $K$-dimensional topic-specific communication patterns.

In the next section, we present the mathematical details of our new model and outline a corresponding inference algorithm. We then introduce a new email data set: the New Hanover County (NHC) email network. Although our model is intended for exploratory analysis, we test our modeling assumptions via three validation tasks. In Section 4.1, we show that our model achieves better link prediction performance than three state-of-the-art network models. We also demonstrate that our model is capable of inferring topics that are as coherent as those inferred using LDA. Together, these experiments indicate that our model is an appropriate model of network structure and that modeling this structure does not compromise topic quality. As a final validation experiment, we show that synthetic data generated using our model possesses similar network statistics to those of the NHC email network. In Section 4.4, we showcase our model's ability to discover and visualize topic-specific communication patterns using the NHC network. We give an extensive analysis of these communication patterns and demonstrate that they provide accessible visualizations of email-based collaboration while possessing precise, meaningful interpretations within the mathematical framework of our model. These findings lead us to recommend our model for any exploratory analysis of email networks or other similarly-structured communication data. Finally, we advocate for principled visualization as a primary objective in the development of new network models.

## 2   Topic-Partitioned Multinetwork Embeddings

In this section, we present our new probabilistic generative model (and associated inference algorithm) for communication networks. For concreteness, we frame our discussion of this model in

terms of email data, although it is generally applicable to any similarly-structured communication data. The generative process and graphical model are provided in the supplementary materials.

A single email, indexed by $d$, is represented by a set of tokens $\boldsymbol{w}^{(d)} = \{w_n^{(d)}\}_{n=1}^{N^{(d)}}$ that comprise the text of that email, an integer $a^{(d)} \in \{1, ..., A\}$ indicating the identity of that email's author, and a set of binary variables $\boldsymbol{y}^{(d)} = \{y_r^{(d)}\}_{r=1}^A$ indicating whether each of the $A$ actors in the network is a recipient of that email. For simplicity, we assume that authors do not send emails to themselves (i.e., $y_r^{(d)} = 0$ if $r = a^{(d)}$). Given a real-world email data set $\mathcal{D} = \{\{\boldsymbol{w}^{(d)}, a^{(d)}, \boldsymbol{y}^{(d)}\}\}_{d=1}^D$, our model permits inference of the topics expressed in the text of the emails, a set of topic-specific $K$-dimensional embeddings (i.e., points in $K$-dimensional Euclidean space) of the $A$ actors in the network, and a partition of the full communication network into a set of topic-specific subnetworks.

As in LDA [6], a "topic" $t$ is characterized by a discrete distribution over $V$ word types with probability vector $\boldsymbol{\phi}^{(t)}$. A symmetric Dirichlet prior with concentration parameter $\beta$ is placed over $\Phi = \{\boldsymbol{\phi}^{(1)}, ..., \boldsymbol{\phi}^{(T)}\}$. To capture the relationship between the topics expressed in an email and that email's recipients, each topic $t$ is also associated with a "communication pattern": an $A \times A$ matrix of probabilities $\boldsymbol{P}^{(t)}$. Given an email about topic $t$, authored by actor $a$, element $p_{ar}^{(t)}$ is the probability of actor $a$ including actor $r$ as a recipient of that email. Inspired by LSM [7], each communication pattern $\boldsymbol{P}^{(t)}$ is represented implicitly via a set of $A$ points in $K$-dimensional Euclidean space $\boldsymbol{S}^{(t)} = \{\boldsymbol{s}_a^{(t)}\}_{a=1}^A$ and a scalar bias term $b^{(t)}$ such that $p_{ar}^{(t)} = p_{ra}^{(t)} = \sigma(b^{(t)} - \|\boldsymbol{s}_a^{(t)} - \boldsymbol{s}_r^{(t)}\|)$ with $\boldsymbol{s}_a^{(t)} \sim \mathcal{N}(\boldsymbol{0}, \sigma_1^2 \boldsymbol{I})$ and $b^{(t)} \sim \mathcal{N}(\mu, \sigma_2^2)$.[1] If $K = 2$ or $K = 3$, this representation enables each topic-specific communication pattern to be visualized in 2- or 3-dimensional space via the locations of the points associated with the $A$ actors. It is worth noting that the dimensions of each $K$-dimensional space have no inherent meaning. In isolation, each point $\boldsymbol{s}_a^{(t)}$ conveys no information; however, the distance between any two points has a precise and meaningful interpretation in the generative process. Specifically, the recipients of any email associated with topic $t$ are more likely to be those actors near to the email's author in the Euclidean space corresponding to that topic.

Each email, indexed by $d$, has a discrete distribution over topics $\boldsymbol{\theta}^{(d)}$. A symmetric Dirichlet prior with concentration parameter $\alpha$ is placed over $\Theta = \{\boldsymbol{\theta}^{(1)}, ..., \boldsymbol{\theta}^{(D)}\}$. Each token $w_n^{(d)}$ is associated with a topic assignment $z_n^{(d)}$, such that $z_n^{(d)} \sim \boldsymbol{\theta}^{(d)}$ and $w_n^{(d)} \sim \boldsymbol{\phi}^{(t)}$ for $z_n^{(d)} = t$. Our model does not include a distribution over authors; the generative process is conditioned upon their identities. The email-specific binary variables $\boldsymbol{y}^{(d)} = \{y_r^{(d)}\}_{r=1}^A$ indicate the recipients of email $d$ and thus the presence (or absence) of email-specific edges from author $a^{(d)}$ to each of the $A - 1$ other actors. Consequently, there may be multiple edges (one per email) between any pair of actors, and $\mathcal{D}$ defines a multinetwork over the entire set of actors. We assume that the complete multinetwork comprises $T$ topic-specific subnetworks. In other words, each $y_r^{(d)}$ is associated with some topic $t$ and therefore with topic-specific communication pattern $\boldsymbol{P}^{(t)}$ such that $y_r^{(d)} \sim \mathrm{Bern}(p_{ar}^{(t)})$ for $a^{(d)} = a$. A natural way to associate each $y_r^{(d)}$ with a topic would be to draw a topic assignment from $\boldsymbol{\theta}^{(d)}$ in a manner analogous to the generation of $z_n^{(d)}$; however, as outlined by Blei and Jordan [8], this approach can result in the undesirable scenario in which one subset of topics is associated with tokens, while another (disjoint) subset is associated with edges. Additionally, models of annotated data that possess this exchangeable structure tend to exhibit poor generalization [3, 8]. A better approach, advocated by Blei and Jordan, is to draw a topic assignment for each $y_r^{(d)}$ from the empirical distribution over topics defined by $\boldsymbol{z}^{(d)}$. By definition, the set of topics associated with edges will therefore be a subset of the topics associated with tokens. One way of simulating this generative process is to associate each $y_r^{(d)}$ with a position $n = 1, \ldots, \max(1, N^{(d)})$ and therefore with the topic assignment $z_n^{(d)}$ at that position[2] by drawing a position assignment $x_r^{(d)} \sim \mathrm{U}(1, \ldots, \max(1, N^{(d)}))$ for each $y_r^{(d)}$. This indirect procedure ensures that $y_r^{(d)} \sim \mathrm{Bern}(p_{ar}^{(t)})$ for $a^{(d)} = a$, $x_r^{(d)} = n$, and $z_n^{(d)} = t$, as desired.

## 2.1 Inference

For real-world data $\mathcal{D} = \{\boldsymbol{w}^{(d)}, a^{(d)}, \boldsymbol{y}^{(d)}\}_{d=1}^{D}$, the tokens $\mathcal{W} = \{\boldsymbol{w}^{(d)}\}_{d=1}^{D}$, authors $\mathcal{A} = \{a^{(d)}\}_{d=1}^{D}$, and recipients $\mathcal{Y} = \{\boldsymbol{y}^{(d)}\}_{d=1}^{D}$ are observed, while $\Phi, \Theta, \mathcal{S} = \{\boldsymbol{S}^{(t)}\}_{t=1}^{T}, \mathcal{B} = \{b^{(t)}\}_{t=1}^{T}, \mathcal{Z} = \{\boldsymbol{z}^{(d)}\}_{d=1}^{D}$, and $\mathcal{X} = \{\boldsymbol{x}^{(d)}\}_{d=1}^{D}$ are unobserved. Dirichlet–multinomial conjugacy allows $\Phi$ and $\Theta$ to be marginalized out [9], while typical values for the remaining unobserved variables can be sampled from their joint posterior distribution using Markov chain Monte Carlo methods. In this section, we outline a Metropolis-within-Gibbs sampling algorithm that operates by sequentially resampling the value of each latent variable (i.e., $\boldsymbol{s}_a^{(t)}$, $b_t$, $z_n^{(d)}$, or $x_r^{(d)}$) from its conditional posterior.

Since $z_n^{(d)}$ is a discrete random variable, new values may be sampled directly using

$$P(z_n^{(d)}\!=\!t \mid w_n^{(d)}\!=\!v, \mathcal{W}_{\backslash d,n}, \mathcal{A}, \mathcal{Y}, \mathcal{S}, \mathcal{B}, \mathcal{Z}_{\backslash d,n}, \mathcal{X}, \alpha, \beta)$$

$$\propto \begin{cases} (N_{\backslash d,n}^{(t|d)} + \frac{\alpha}{T}) \frac{N_{\backslash d,n}^{(v|t)} + \frac{\beta}{V}}{N_{\backslash d,n}^{(t)} + \beta} \prod_{r:x_r^{(d)}=n} \left(p_{a^{(d)}r}^{(t)}\right)^{y_r^{(d)}} \left(1 - p_{a^{(d)}r}^{(t)}\right)^{1-y_r^{(d)}} & \text{for } N^{(d)} > 0 \\ \prod_{r:r\neq a^{(d)}} \left(p_{a^{(d)}r}^{(t)}\right)^{y_r^{(d)}} \left(1 - p_{a^{(d)}r}^{(t)}\right)^{1-y_r^{(d)}} & \text{otherwise,} \end{cases}$$

where subscript "$\backslash d, n$" denotes a quantity excluding data from position $n$ in email $d$. Count $N^{(t)}$ is the total number of tokens in $\mathcal{W}$ assigned to topic $t$ by $\mathcal{Z}$, of which $N^{(v|t)}$ are of type $v$ and $N^{(t|d)}$ belong to email $d$. New values for discrete random variable $x_r^{(d)}$ may be sampled directly using

$$P(x_r^{(d)}\!=\!n \mid \mathcal{A}, \mathcal{Y}, \mathcal{S}, \mathcal{B}, z_n^{(d)}\!=\!t, \mathcal{Z}_{\backslash d,n}) \propto \left(p_{a^{(d)}r}^{(t)}\right)^{y_r^{(d)}} \left(1 - p_{a^{(d)}r}^{(t)}\right)^{1-y_r^{(d)}}.$$

New values for continuous random variables $\boldsymbol{s}_a^{(t)}$ and $b^{(t)}$ cannot be sampled directly from their conditional posteriors, but may instead be obtained using the Metropolis–Hastings algorithm. With a non-informative prior over $\boldsymbol{s}_a^{(t)}$ (i.e., $\boldsymbol{s}_a^{(t)} \sim \mathcal{N}(\boldsymbol{0}, \infty)$), the conditional posterior over $\boldsymbol{s}_a^{(t)}$ is

$$P(\boldsymbol{s}_a^{(t)} \mid \mathcal{A}, \mathcal{Y}, \boldsymbol{S}_{\backslash a}^{(t)}, b^{(t)}, \mathcal{Z}, \mathcal{X}) \propto \prod_{r:r\neq a} \left(p_{ar}^{(t)}\right)^{N^{(1|a,r,t)}+N^{(1|r,a,t)}} \left(1 - p_{ar}^{(t)}\right)^{N^{(0|a,r,t)}+N^{(0|r,a,t)}},$$

where count $N^{(1|a,r,t)} = \sum_{d=1}^{D} \mathbf{1}(a^{(d)}\!=\!a) \, \mathbf{1}(y_r^{(d)}\!=\!1) \left(\sum_{n=1}^{N^{(d)}} \mathbf{1}(x_r^{(d)}\!=\!n) \, \mathbf{1}(z_n^{(d)}\!=\!t)\right)$.[3] Counts $N^{(1|r,a,t)}$, $N^{(0|a,r,t)}$, and $N^{(0|r,a,t)}$ are defined similarly. Likewise, with an improper, non-informative prior over $b^{(t)}$ (i.e., $b^{(t)} \sim \mathcal{N}(0, \infty)$), the conditional posterior over $b^{(t)}$ is

$$P(b^{(t)} \mid \mathcal{A}, \mathcal{Y}, \boldsymbol{S}^{(t)}, \mathcal{Z}, \mathcal{X}) \propto \prod_{a=1}^{A} \prod_{r:r<a} \left(p_{ar}^{(t)}\right)^{N^{(1|a,r,t)}+N^{(1|r,a,t)}} \left(1 - p_{ar}^{(t)}\right)^{N^{(0|a,r,t)}+N^{(0|r,a,t)}}.$$

## 3 Data

Due to a variety of factors involving personal privacy concerns and the ownership of content by email service providers, academic researchers rarely have access to organizational email data. For example, the Enron data set [10]—arguably the most widely studied email data set—was only released because of a court order. The public record is an alternative source of organizational email data. Public record data sets are widely available and can be continually updated, yet remain relatively untapped by the academic community. We therefore introduce and analyze a new public record email data set relevant to researchers in the organizational and social sciences as well as machine learning researchers. This data set consists of emails between the managers of the departments that constitute the executive arm of government at the county level for New Hanover County, North Carolina. In this semi-autonomous local government, county managers act as executives, and the individual departments are synonymous with the individual departments and agencies in, for instance, the U.S. federal government. Therefore, not only does this email data set offer a view into the communication patterns of the managers of New Hanover County, but analyses of it also serve as case studies in modeling inter-agency communications in the U.S. federal government administration.

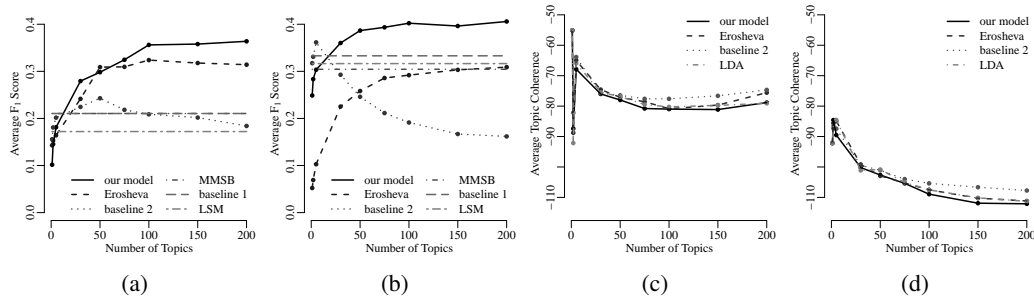

Figure 2: Average link prediction performance for (a) the NHC email network and (b) the Enron data set. For MMSB and LSM, we only report results obtained using the best-performing hyperparameter values. Average topic coherence scores for (c) the NHC email network and (d) the Enron data set.

The New Hanover County (NHC) email network comprises the complete inboxes and outboxes of 30 department managers from the month of February, 2011. In total, there are 30,909 emails, of which 8,097 were authored by managers. Of these 8,097 emails, 1,739 were sent to other managers (via the "To" or "Cc" fields), excluding any emails sent from a manager to him- or herself only. For our experiments, we used these 1,739 emails between 30 actors. To verify that our model is applicable beyond the NHC email network, we also performed two validation experiments using the Enron email data set [10]. For this data set, we treated each unique @enron email address as an actor and used only those emails between the 50 most active actors (determined by the total numbers of emails sent and received). Emails that were not sent to at least one other active actor (via the "To" or "Cc" fields) were discarded. To avoid duplicate emails, we retained only those emails from "_sent_mail", "sent", or "sent_items" folders. These steps resulted in a total of 8,322 emails involving 50 actors. Both data sets were preprocessed to concatenate the text of subject lines and message bodies and to remove any stop words, URLs, quoted text, and (where possible) signatures.

## 4 Experiments

Our model is primarily intended as an exploratory analysis tool for organizational communication networks. In this section, we use the NHC email network to showcase our model's ability to discover and visualize topic-specific communication subnetworks. First, however, we test our underlying modeling assumptions via three quantitative validation tasks, as recommended by Schrodt [11].

### 4.1 Link Prediction

In order to gauge our model's predictive performance, we evaluated its ability to predict the recipients of "test" emails, from either the NHC email network or the Enron data set, conditioned on the text of those emails and the identities of their authors. For each test email $d$, the binary variables indicating the recipients of that email, i.e., $\{y_r^{(d)}\}_{r=1}^{A}$, were treated as unobserved. Typical values for these variables were sampled from their joint posterior distribution and compared to the true values to yield an $F_1$ score. We formed a test split of each data set by randomly selecting emails with probability 0.1. For each data set, we averaged the $F_1$ scores over five random test splits.

We compared our model's performance with that of two baselines and three existing network models, thereby situating it within the existing literature. Given a test email authored by actor $a$, our simplest baseline naïvely predicts that actor $a$ will include actor $r$ as a recipient of that email with probability equal to the number of non-test emails sent from actor $a$ to actor $r$ divided by the total number of non-test emails sent by actor $a$. Our second baseline is a variant of our model in which each topic-specific communication pattern $\boldsymbol{P}^{(t)}$ is represented explicitly via $A(A+1)/2$ probabilities drawn from a symmetric Beta prior with concentration parameter $\gamma$. Comparing our model to this variant enables us to validate our assumption that topic-specific communication patterns can indeed be accurately represented by a set of $A$ points (one per actor) in $K$-dimensional Euclidean space. We also compared our model's performance to that of three existing network models: a variant of Erosheva et al.'s model for analyzing scientific publications [4], LSM [7], and the

mixed-membership stochastic blockmodel (MMSB) [12]. Erosheva et al.'s model can be viewed as a variant of our model in which the topic assignment for each $y_r^{(d)}$ is drawn from $\boldsymbol{\theta}^{(d)}$ instead of the empirical distribution over topics defined by $\boldsymbol{z}^{(d)}$. Like our second baseline, each topic-specific communication pattern is represented explicitly via probabilities drawn from a symmetric Beta prior with concentration parameter $\gamma$; however, unlike this baseline, each one is represented using $A$ probabilities such that $p_{ar}^{(t)} = p_r^{(t)}$. LSM can be viewed as a network-only variant of our model in which text is not modeled. As a result, there are no topics and a single communication pattern $\boldsymbol{P}$. This pattern is represented implicitly via a set of $A$ actor-specific points in $K$-dimensional Euclidean space. Finally, MMSB is a widely-used model for mixed-membership community discovery in networks.

For our model and all its variants, typical values for $\{y_r^{(d)}\}_{r=1}^A$ can be sampled from their joint posterior distribution using an appropriately-modified version of the Metropolis-within-Gibbs algorithm in Section 2.1. In all our experiments, we ran this algorithm for 40,000–50,000 iterations. On iteration $i$, we defined each proposal distribution to be a Gaussian distribution centered on the value from iteration $i-1$ with covariance matrix $\max\left(1, 100\,/\,i\right)\mathbf{I}$, thereby resulting in larger covariances for earlier iterations. Beta–binomial conjugacy allows the elements of $\boldsymbol{P}^{(t)}$ to be marginalized out in both our second baseline and Erosheva et al.'s model. For MMSB, typical values can be sampled using a modified version of Chang's Gibbs sampling algorithm [13]. We ran this algorithm for 5,000 iterations. For all models involving topics, we set concentration parameter $\alpha$ to 1 for the NHC network and 2 for the Enron data set. For both data sets, we set concentration parameter $\beta$ to $0.01V$.[4] We varied the number of topics from 1 to 200. In order to facilitate visualization, we used 2-dimensional Euclidean spaces for our model. For LSM, however, we varied the dimensionality of the Euclidean space from 1 to 200. We report only those results obtained using the best-performing dimensionality. For our second baseline and Erosheva et al.'s model, we set concentration parameter $\gamma$ to 0.02. For MMSB, we performed a grid search over all hyperparameter values and the number of blocks and, as with LSM, report only those results obtained using the best-performing values.[5]

$F_1$ scores, averaged over five random test splits of each data set, are shown in Figure 2. Although our model is intended for exploratory analysis, it achieves better link prediction performance than the other models. Furthermore, the fact that our model outperforms our second baseline and Erosheva et al.'s model validates our assumption that topic-specific communication patterns can indeed be accurately represented by a set of $A$ actor-specific points in 2-dimensional Euclidean space.

## 4.2 Topic Coherence

When evaluating unsupervised topic models, topic coherence metrics [14, 15] are often used as a proxy for subjective evaluation of semantic coherence. In order to demonstrate that incorporating network data does not impair our model's ability to model text, we compared the coherence of topics inferred using our model with the coherence of topics inferred using LDA, our second baseline, and Erosheva et al.'s model. For each model, we varied the number of topics from 1 to 200 and drew five samples from the joint posterior distribution over the unobserved random variables in that model. We evaluated the topics resulting from each sample using Mimno et al.'s coherence metric [14]. Topic coherence, averaged over the five samples, is shown in Figure 2. Our model achieves coherence comparable to that of LDA. This result, when combined with the results in Section 4.1, demonstrates that our model can achieve state-of-the-art predictive performance while producing coherent topics.

## 4.3 Posterior Predictive Checks

We used posterior predictive checking to assess the extent to which our model is a "good fit" for the NHC email network [16, 17]. Specifically, we defined four network statistics (i.e., four discrepancy functions) that summarize meaningful aspects of the NHC network: generalized graph transitivity, the dyad intensity distribution, the vertex degree distribution, and the geodesic distance distribution.[6] We then generated 1,000 synthetic networks from the posterior predictive distribution implied by our

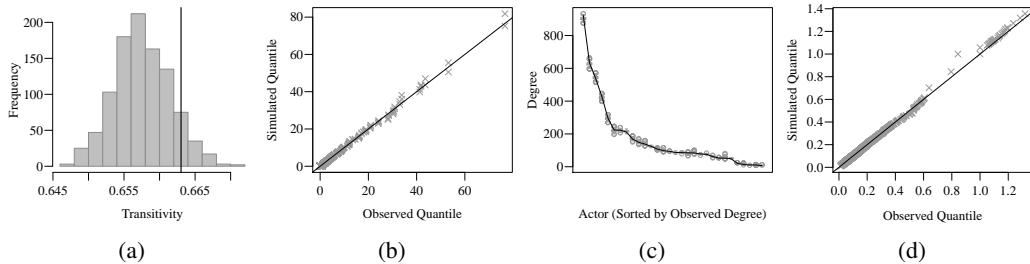

Figure 3: Four posterior predictive checks of our model using the NHC email network and 100 topics: (a) a histogram of the graph transitivity of the synthetic networks, with the graph transitivity of the NHC email network indicated by a vertical line; (b) a quantile–quantile plot comparing the distribution of dyadic intensities in the synthetic networks to that of the observed network; (c) a box plot indicating the sampled degree of each manager in the synthetic networks, with managers sorted from highest to lowest observed degree and their observed degrees indicated by a line; and (d) a quantile–quantile plot comparing the observed and synthetic geodesic distance distributions.

model and the NHC network. We applied each discrepancy function to each synthetic network to yield four distributions over the values of the four network statistics. If our model is a "good fit" for the NHC network, these distributions should be centered around the values of the corresponding discrepancy functions when computed using the observed NHC network. As shown in Figure 3, our model generates synthetic networks with dyad intensity, vertex degree, and geodesic distance distributions that are very similar to those of the NHC network. The distribution over synthetic graph transitivity values is not centered around the observed graph transitivity, but the observed transitivity is not sufficiently far into the tail of the distribution to warrant reparameterization of our model.

## 4.4  Exploratory Analysis

In order to demonstrate our model's novel ability to discover and visualize topic-specific communication patterns, we performed an exploratory analysis of four such patterns inferred from the NHC email network using our model. These patterns are visualized in Figure 4. Each pattern is represented implicitly via a single set of $A$ points in 2-dimensional Euclidean space drawn from their joint posterior distribution. The recipients of any email associated with topic $t$ are more likely to be those actors near to the email's author in the Euclidean space corresponding to that topic. We selected the patterns in Figure 4 so as to highlight the types of insights that can be obtained using our model. Although many structural properties may be of interest, we focus on modularity and assortativity.

For each topic-specific communication pattern, we examined whether there are active, disconnected components in that topic's Euclidean space (i.e., high modularity). The presence of such components indicates that there are groups of actors who engage in within- but not between-group communication about that topic. We also used a combination of node proximity and node coloration to determine whether there is more communication between departments that belong to the same "division" in the New Hanover County government organizational chart than between departments within different divisions (i.e., assortativity). In Figure 4, we show one topic that exhibits strong modularity and little assortativity (the "Public Signage" topic), one topic that exhibits strong assortativity and little modularity (the "Broadcast Messages" topic), and one topic that exhibits both strong assortativity and strong modularity (the "Meeting Scheduling" topic). The "Public Relations" topic, which includes communication with news agencies, is mostly dominated by a cluster involving many departments. Finally, the "Meeting Scheduling" topic displays hierarchical structure, with two assistant county managers located at the centers of groups that correspond to their divisions.

Exploratory analysis of communication patterns is a powerful tool for understanding organizational communication networks. For example, examining assortativity can reveal whether actual communication patterns resemble official organizational structures. Similarly, if a communication pattern exhibits modularity, each disconnected component may benefit from organizational efforts to facilitate inter-component communication. Finally, structural properties other than assortativity and modularity may also yield scientific or practical insights, depending on organizational needs.

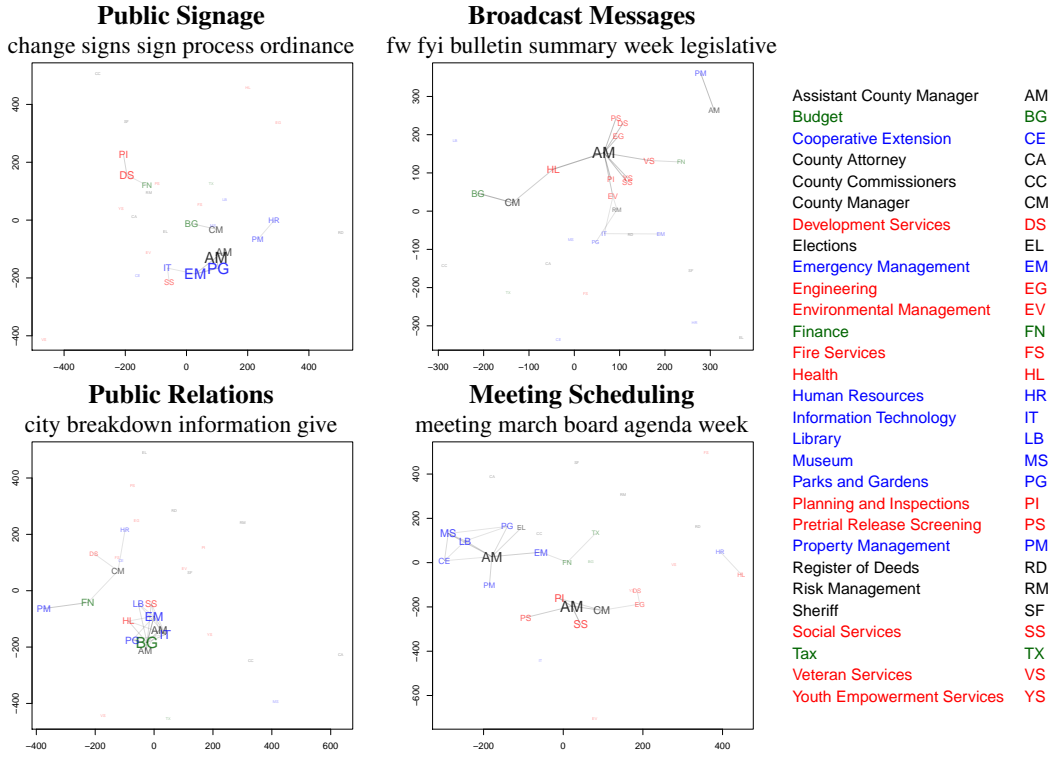

Figure 4: Four topic-specific communication patterns inferred from the NHC email network. Each pattern is labeled with a human-selected name for the corresponding topic, along with that topic's most probable words in order of decreasing probability. The size of each manager's acronym in topic $t$'s pattern (given by $0.45 + 1.25\sqrt{d_a^{(t)} / \max_a d_a^{(t)}}$, where $d_a^{(t)}$ is the degree of actor $a$ in that subnetwork) indicates how often that manager communicates about that topic. Managers' acronyms are colored according to their respective division in the New Hanover County organizational chart. The acronym "AM" appears twice in all plots because there are two assistant county managers.

## 5    Conclusions

We introduced a new Bayesian admixture model for the discovery and visualization of topic-specific communication subnetworks. Although our model is intended for exploratory analysis, the validation experiments described in Sections 4.1 and 4.2 demonstrate that our model can achieve state-of-the-art predictive performance while exhibiting topic coherence comparable to that of LDA. To showcase our model's ability to discover and visualize topic-specific communication patterns, we introduced a new data set (the NHC email network) and analyzed four such patterns inferred from this data set using our model. Via this analysis, were are able to examine the extent to which actual communication patterns resemble official organizational structures and identify groups of managers who engage in within- but not between-group communication about certain topics. Together, these predictive and exploratory analyses lead us to recommend our model for any exploratory analysis of email networks or other similarly-structured communication data. Finally, our model is capable of producing principled visualizations of email networks, i.e., visualizations that have precise mathematical interpretations in terms of this model and its relationship to the observed data. We advocate for principled visualization as a primary objective in the development of new network models.

## Acknowledgments

This work was supported in part by the Center for Intelligent Information Retrieval and in part by the NSF GRFP under grant #1122374. Any opinions, findings, and conclusions or recommendations expressed in this material are those of the authors and do not necessarily reflect those of the sponsors.

## Footnotes

[1] The function $\sigma(\cdot)$ is the logistic function, while the function $\| \cdot \|$ is the $l^2$-norm.

[2] Emails that do not contain any text (i.e., $N^{(d)} = 0$) convey information about the frequencies of communication between their authors and recipients. As a result, we do not omit such emails from $\mathcal{D}$; instead, we augment each one with a single, "dummy" topic assignment $z_1^{(d)}$ for which there is no associated token $w_1^{(d)}$.

[3]The function $\mathbf{1}(\cdot)$ evaluates to one if its argument evaluates to true and evaluates to zero otherwise.

[4]These values were obtained by slice sampling typical values for the concentration parameters in LDA. They are consistent with the concentration parameter values used in previous work [9].

[5]These values correspond to a $\mathrm{Dir}(0.1,\ldots,0.1)$ prior over block memberships, a $\mathrm{Beta}(0.1,0.1)$ prior over diagonal entries of the blockmodel, a $\mathrm{Beta}(0.01,0.01)$ prior over off-diagonal entries, and 30 blocks.

[6]These statistics are defined in the supplementary materials.

# References

[1] W. Mason and D.J. Watts. Collaborative learning in networks. *Proceedings of the National Academy of Sciences*, 109(3):764–769, 2012.

[2] A. McCallum, A. Corrada-Emmanuel, and X. Wang. Topic and role discovery in social networks. In *Proceedings of the International Joint Conference on Artificial Intelligence*, 2005.

[3] J. Chang and D.M. Blei. Relational topic models for document networks. In *Proceedings of the Twelfth International Conference on Artificial Intelligence and Statistics*, 2009.

[4] E. Erosheva, S. Fienberg, and J. Lafferty. Mixed-membership models of scientific publications. *Proceedings of the National Academy of Sciences*, 101(Suppl. 1), 2004.

[5] S.E Fienberg. A brief history of statistical models for network analysis and open challenges. *Journal of Computational and Graphical Statistics*, 22, 2012.

[6] D.M. Blei, A.Y. Ng, and M.I. Jordan. Latent Dirichlet allocation. *Journal of Machine Learning Research*, 3:993–1022, 2003.

[7] P.D. Hoff, A.E. Raftery, and M.S. Handcock. Latent space approaches to social network analysis. *Journal of the American Statistical Association*, 97(460):1090–1098, 2002.

[8] D.M. Blei and M.I. Jordan. Modeling annotated data. In *Proceedings of the Twenty-Sixth Annual International ACM SIGIR Conference on Research and Development in Information Retrieval*, pages 127–134, 2003.

[9] T.L. Griffiths and M. Steyvers. Finding scientific topics. *Proceedings of the National Academy of Sciences*, 101(Suppl. 1), 2004.

[10] B. Klimt and Y. Yang. Introducing the Enron corpus. In *Proceedings of the First Conference on Email and Anti-Spam*, 2004.

[11] P.A Schrodt. Seven deadly sins of contemporary quantitative political analysis. In *Proceedings of the Annual American Political Science Association Meeting and Exhibition*, 2010.

[12] E.M. Airoldi, D.M. Blei, S.E. Fienberg, and E.P. Xing. Mixed membership stochastic block-models. *Journal of Machine Learning Research*, 9:1981–2014, 2008.

[13] J. Chang. *Uncovering, Understanding, and Predicting Links*. PhD thesis, Princeton Unversity, 2011.

[14] D. Mimno, H.M. Wallach, E.T.M. Leenders, and A. McCallum. Optimizing semantic coherence in topic models. In *Proceedings of the Conference on Empirical Methods in Natural Language Processing*, 2011.

[15] D. Newman, J.H. Lau, K. Grieser, and T. Baldwin. Automatic evaluation of topic coherence. In *Proceedings of Human Language Technologies: The Annual Conference of the North American Chapter of the Association for Computational Linguistics*, pages 100–108, 2010.

[16] D.R. Hunter, M.S. Handcock, C.T. Butts, S.M. Goodreau, and M. Morris. ergm: A package to fit, simulate and diagnose exponential-family models for networks. *Journal of Statistical Software*, 24(3):1–29, 2008.

[17] D. Mimno and D.M. Blei. Bayesian checking for topic models. In *Proceedings of the Conference on Empirical Methods in Natural Language Processing*, pages 227–237, 2011.

